# Iterative Construction of Sparse Polynomial Approximations

**Terence D. Sanger**
Massachusetts Institute
of Technology
Room E25-534
Cambridge, MA 02139
tds@ai.mit.edu

**Richard S. Sutton**
GTE Laboratories
Incorporated
40 Sylvan Road
Waltham, MA 02254
sutton@gte.com

**Christopher J. Matheus**
GTE Laboratories
Incorporated
40 Sylvan Road
Waltham, MA 02254
matheus@gte.com

## Abstract

We present an iterative algorithm for nonlinear regression based on construction of sparse polynomials. Polynomials are built sequentially from lower to higher order. Selection of new terms is accomplished using a novel look-ahead approach that predicts whether a variable contributes to the remaining error. The algorithm is based on the tree-growing heuristic in LMS Trees which we have extended to approximation of arbitrary polynomials of the input features. In addition, we provide a new theoretical justification for this heuristic approach. The algorithm is shown to discover a known polynomial from samples, and to make accurate estimates of pixel values in an image-processing task.

## 1 INTRODUCTION

Linear regression attempts to approximate a target function by a model that is a linear combination of the input features. Its approximation ability is thus limited by the available features. We describe a method for adding new features that are products or powers of existing features. Repeated addition of new features leads to the construction of a polynomial in the original inputs, as in (Gabor 1961). Because there is an infinite number of possible product terms, we have developed a new method for predicting the usefulness of entire classes of features before they are included. The resulting nonlinear regression will be useful for approximating functions that can be described by sparse polynomials.

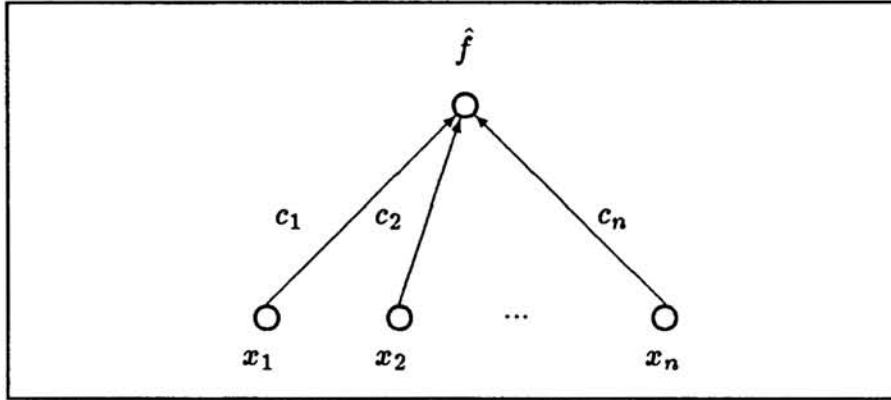

Figure 1: Network depiction of linear regression on a set of features $x_i$.

## 2  THEORY

Let $\{x_i\}_{i=1}^{n}$ be the set of features already included in a model that attempts to predict the function $f$. The output of the model is a linear combination

$$\hat{f} = \sum_{i=1}^{n} c_i x_i$$

where the $c_i$'s are coefficients determined using linear regression. The model can also be depicted as a single-layer network as in figure 1. The approximation error is $e = f - \hat{f}$, and we will attempt to minimize $E[e^2]$ where $E$ is the expectation operator.

The algorithm incrementally creates new features that are products of existing features. At each step, the goal is to select two features $x_p$ and $x_q$ already in the model and create a new feature $x_p x_q$ (see figure 2). Even if $x_p x_q$ does not decrease the approximation error, it is still possible that $x_p x_q x_r$ will decrease it for some $x_r$. So in order to decide whether to create a new feature that is a product with $x_p$, the algorithm must "look-ahead" to determine if there exists any polynomial $a$ in the $x_i$'s such that inclusion of $a x_p$ would significantly decrease the error. If no such polynomial exists, then we do not need to consider adding any features that are products with $x_p$.

Define the inner product between two polynomials $a$ and $b$ as $\langle a|b \rangle = E[ab]$ where the expected value is taken with respect to a probability measure $\mu$ over the (zero-mean) input values. The induced norm is $\|a\|^2 = E[a^2]$, and let $P$ be the set of polynomials with finite norm. $\{P, \langle \cdot | \cdot \rangle\}$ is then an infinite-dimensional linear vector space. The Weierstrass approximation theorem proves that $P$ is dense in the set of all square-integrable functions over $\mu$, and thus justifies the assumption that any function of interest can be approximated by a member of $P$.

Assume that the error $e$ is a polynomial in $P$. In order to test whether $a x_p$ participates in $e$ for any polynomial $a \in P$, we write

$$e = a_p x_p + b_p$$

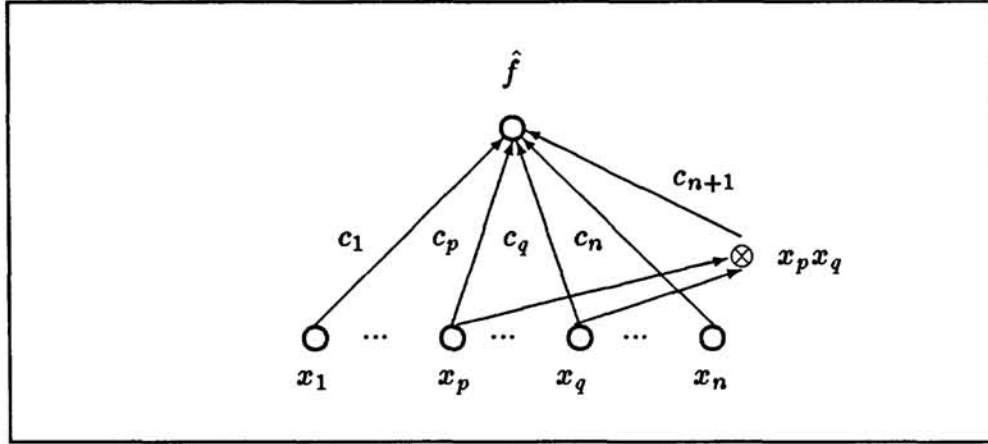

Figure 2: Incorporation of a new product term into the model.

where $a_p$ and $b_p$ are polynomials, and $a_p$ is chosen to minimize $\|a_p x_p - e\|^2 = E[(a_p x_p - e)^2]$. The orthogonality principle then shows that $a_p x_p$ is the projection of the polynomial $e$ onto the linear subspace of polynomials $x_p P$. Therefore, $b_p$ is orthogonal to $x_p P$, so that $E[b_p g] = 0$ for all $g$ in $x_p P$.

We now write

$$E[e^2] = E[a_p^2 x_p^2] + 2E[a_p x_p b_p] + E[b_p^2] = E[a_p^2 x_p^2] + E[b_p^2]$$

since $E[a_p x_p b_p] = 0$ by orthogonality. If $a_p x_p$ were included in the model, it would thus reduce $E[e^2]$ by $E[a_p^2 x_p^2]$, so we wish to choose $x_p$ to maximize $E[a_p^2 x_p^2]$. Unfortunately, we have no direct measurement of $a_p$.

## 3   METHODS

Although $E[a_p^2 x_p^2]$ cannot be measured directly, Sanger (1991) suggests choosing $x_p$ to maximize $E[e^2 x_p^2]$ instead, which is directly measurable. Moreover, note that

$$\begin{aligned} E[e^2 x_p^2] &= E[a_p^2 x_p^4] + 2E[a_p x_p^3 b_p] + E[x_p^2 b_p^2] \\ &= E[a_p^2 x_p^4] \end{aligned}$$

and thus $E[e^2 x_p^2]$ is related to the desired but unknown value $E[a_p^2 x_p^2]$. Perhaps better would be to use

$$\frac{E[e^2 x_p^2]}{E[x_p^2]} = \frac{E[a_p^2 x_p^4]}{E[x_p^2]}$$

which can be thought of as the regression of $(a_p^2 x_p^2) x_p$ against $x_p$.

More recently, (Sutton and Matheus 1991) suggest using the regression coefficients of $e^2$ against $x_i^2$ for all $i$ as the basis for comparison. The regression coefficients $w_i$ are called "potentials", and lead to a linear approximation of the squared error:

$$\widehat{e^2} = \sum_{i=1}^n w_i x_i^2 \tag{1}$$

If a new term $a_p x_p$ were included in the model of $f$, then the squared error would be $b_p^2$ which is orthogonal to any polynomial in $x_p P$ and in particular to $x_p^2$. Thus the coefficient of $x_p^2$ in (1) would be zero after inclusion of $a_p x_p$, and $w_p E[x_p^2]$ is an approximation to the decrease in mean-squared error $E[e^2] - E[b_p^2]$ which we can expect from inclusion of $a_p x_p$. We thus choose $x_p$ by maximizing $w_p E[x_p^2]$.

This procedure is a form of look-ahead which allows us to predict the utility of a high-order term $a_p x_p$ without actually including it in the regression. This is perhaps most useful when the term is predicted to make only a small contribution for the optimal $a_p$, because in this case we can drop from consideration any new features that include $x_p$.

We can choose a different variable $x_q$ similarly, and test the usefulness of incorporating the product $x_p x_q$ by computing a "joint potential" $w_{pq}$ which is the regression of the squared error against the model including a new term $x_p^2 x_q^2$. The joint potential attempts to predict the magnitude of the term $E[a_{pq}^2 x_p^2 x_q^2]$.

We now use this method to choose a single new feature $x_p x_q$ to include in the model. For all pairs $x_i x_j$ such that $x_i$ and $x_j$ individually have high potentials, we perform a third regression to determine the joint potentials of the product terms $x_i x_j$. Any term with a high joint potential is likely to participate in $f$. We choose to include the new term $x_p x_q$ with the largest joint potential. In the network model, this results in the construction of a new unit that computes the product of $x_p$ and $x_q$, as in figure 2. The new unit is incorporated into the regression, and the resulting error $e$ will be orthogonal to this unit and all previous units. Iteration of this technique leads to the successive addition of new regression terms and the successive decrease in mean-squared error $E[e^2]$. The process stops when the residual mean-squared error drops below a chosen threshold, and the final model consists of a sparse polynomial in the original inputs.

We have implemented this algorithm both in a non-iterative version that computes coefficients and potentials based on a fixed data set, and in an iterative version that uses the LMS algorithm (Widrow and Hoff 1960) to compute both coefficients and potentials incrementally in response to continually arriving data. In the iterative version, new terms are added at fixed intervals and are chosen by maximizing over the potentials approximated by the LMS algorithm. The growing polynomial is efficiently represented as a tree-structure, as in (Sanger 1991a).

Although the algorithm involves three separate regressions, each is over only $O(n)$ terms, and thus the iterative version of the algorithm is only of $O(n)$ complexity per input pattern processed.

## 4   RELATION TO OTHER ALGORITHMS

Approximation of functions over a fixed monomial basis is not a new technique (Gabor 1961, for example). However, it performs very poorly for high-dimensional input spaces, since the set of all monomials (even of very low order) can be prohibitively large. This has led to a search for methods which allow the generation of sparse polynomials. A recent example and bibliography are provided in (Grigoriev *et al.* 1990), which describes an algorithm applicable to finite fields (but not to

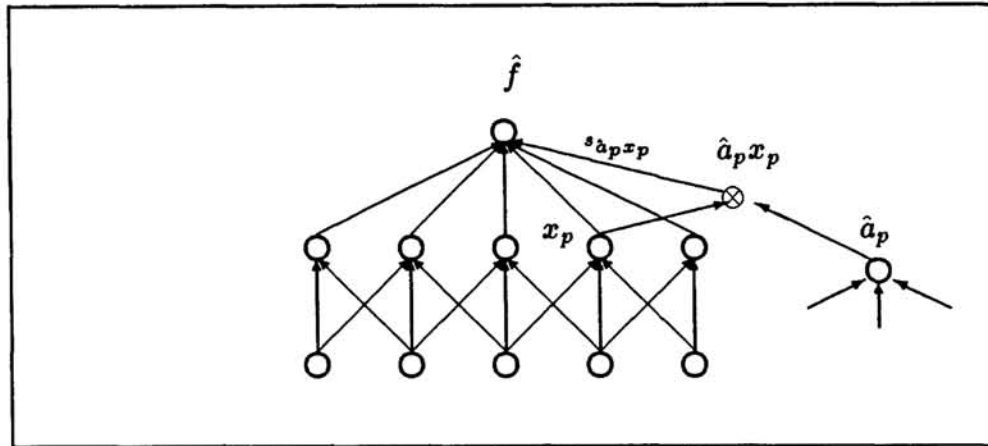

Figure 3: Products of hidden units in a sigmoidal feedforward network lead to a polynomial in the hidden units themselves.

real-valued random variables).

The GMDH algorithm (Ivakhnenko 1971, Ikeda *et al.* 1976, Barron *et al.* 1984) incrementally adds new terms to a polynomial by forming a second (or higher) order polynomial in 2 (or more) of the current terms, and including this polynomial as a new term if it correlates with the error. Since GMDH does not use look-ahead, it risks avoiding terms which would be useful at future steps. For example, if the polynomial to be approximated is $xyz$ where all three variables are independent, then no polynomial in $x$ and $y$ alone will correlate with the error, and thus the term $xy$ may never be included. However, $x^2y^2$ does correlate with $x^2y^2z^2$, so the look-ahead algorithm presented here would include this term, even though the error did not decrease until a later step. Although GMDH can be extended to test polynomials of more than 2 variables, it will always be testing a finite-order polynomial in a finite number of variables, so there will always exist target functions which it will not be able to approximate.

Although look-ahead avoids this problem, it is not always useful. For practical purposes, we may be interested in the best $N$th-order approximation to a function, so it may not be helpful to include terms which participate in monomials of order greater than $N$, even if these monomials would cause a large decrease in error. For example, the best 2nd-order approximation to $x^2 + y^{1000} + z^{1000}$ may be $x^2$, even though the other two terms contribute more to the error. In practice, some combination of both infinite look-ahead and GMDH-type heuristics may be useful.

## 5    APPLICATION TO OTHER STRUCTURES

These methods have a natural application to other network structures. The inputs to the polynomial network can be sinusoids (leading to high-dimensional Fourier representations), Gaussians (leading to high-dimensional Radial Basis Functions) or other appropriate functions (Sanger 1991a, Sanger 1991b). Polynomials can

even be applied with sigmoidal networks as input, so that

$$x_i = \sigma \left( \sum s_{ij} z_j \right)$$

where the $z_j$'s are the original inputs, and the $s_{ij}$'s are the weights to a sigmoidal hidden unit whose value is the polynomial term $x_i$. The last layer of hidden units in a multilayer network is considered to be the set of input features $x_i$ to a linear output unit, and we can compute the potentials of these features to determine the hidden unit $x_p$ that would most decrease the error if $a_p x_p$ were included in the model (for the optimal polynomial $a_p$). But $a_p$ can now be approximated using a subnetwork of any desired type. This subnetwork is used to add a new hidden unit $\hat{a}_p x_p$ that is the product of $x_p$ with the subnetwork output $\hat{a}_p$, as in figure 3.

In order to train the $\hat{a}_p$ subnetwork iteratively using gradient descent, we need to compute the effect of changes in $\hat{a}_p$ on the network error $\mathcal{E} = E[(f - \hat{f})^2]$. We have

$$\frac{\partial \mathcal{E}}{\partial \hat{a}_p} = -2E[(f - \hat{f}) s_{\hat{a}_p x_p} x_p]$$

where $s_{\hat{a}_p x_p}$ is the weight from the new hidden unit to the output. Without loss of generality we can set $s_{\hat{a}_p x_p} = 1$ by including this factor within $\hat{a}_p$. Thus the error term for iteratively training the subnetwork $\hat{a}_p$ is

$$(f - \hat{f}) x_p$$

which can be used to drive a standard backpropagation-type gradient descent algorithm. This gives a method for constructing new hidden nodes and a learning algorithm for training these nodes. The same technique can be applied to deeper layers in a multilayer network.

## 6   EXAMPLES

We have applied the algorithm to approximation of known polynomials in the presence of irrelevant noise variables, and to a simple image-processing task.

Figure 4 shows the results of applying the algorithm to 200 samples of the polynomial $2 + 3x_1 x_2 + 4x_3 x_4 x_5$ with 4 irrelevant noise variables. The algorithm correctly finds the true polynomial in 4 steps, requiring about 5 minutes on a Symbolics Lisp Machine. Note that although the error did not decrease after cycle 1, the term $x_4 x_5$ was incorporated since it would be useful in a later step to reduce the error as part of $x_3 x_4 x_5$ in cycle 2.

The image processing task is to predict a pixel value on the succeeding scan line from a 2x5 block of pixels on the preceding 2 scan lines. If successful, the resulting polynomial can be used as part of a DPCM image coding strategy. The network was trained on random blocks from a single face image, and tested on a different image. Figure 5 shows the original training and test images, the pixel predictions, and remaining error . Figure 6 shows the resulting 55-term polynomial. Learning this polynomial required less than 10 minutes on a Sun Sparcstation 1.

200 samples of $y = 2 + 3x_1 x_2 + 4x_3 x_4 x_5$
with 4 additional irrelevant inputs, $x_6 - x_9$

Original MSE: 1.0

Cycle 1:

| | | $X_1$ | $X_2$ | $X_3$ | $X_4$ | $X_5$ | $X_6$ | $X_7$ | $X_8$ | $X_9$ | | | |
|---|---|---|---|---|---|---|---|---|---|---|---|---|---|
| MSE: | 0.967 | | | | | | | | | | | | |
| Terms: | | $X_1$ | $X_2$ | $X_3$ | $X_4$ | $X_5$ | $X_6$ | $X_7$ | $X_8$ | $X_9$ | | | |
| Coeffs: | | -0.19 | 0.14 | 0.24 | 0.31 | 0.17 | 0.48 | 0.03 | 0.05 | 0.58 | | | |
| Potentials: | | 0.22 | 0.24 | 0.25 | 0.32 | 0.33 | 0.01 | 0.08 | 0.01 | 0.05 | | | |
| Top Pairs: | | (5 4) (5 3) (4 3) (4 4) | | | | | | | | | | | |
| New Term: | | $X_{10} = X_4 X_5$ | | | | | | | | | | | |

Cycle 2:

| | | | | | | | | | | | | |
|---|---|---|---|---|---|---|---|---|---|---|---|---|
| MSE: | 0.966 | | | | | | | | | | | |
| Terms: | | $X_1$ | $X_2$ | $X_3$ | $X_4$ | $X_5$ | $X_6$ | $X_7$ | $X_8$ | $X_9$ | $X_{10}$ | |
| Coeffs: | | -0.19 | 0.14 | 0.24 | 0.30 | 0.18 | 0.48 | 0.03 | 0.05 | 0.57 | 0.05 | |
| Potentials: | | 0.25 | 0.22 | 0.25 | 0.05 | 0.02 | 0.03 | 0.08 | 0.02 | 0.03 | 0.47 | |
| Top Pairs: | | (10 3) (10 1) (10 2) (10 10) | | | | | | | | | | |
| New Term: | | $X_{11} = X_{10} X_3 = X_3 X_4 X_5$ | | | | | | | | | | |

Cycle 3:

| | | | | | | | | | | | | |
|---|---|---|---|---|---|---|---|---|---|---|---|---|
| MSE: | 0.349 | | | | | | | | | | | |
| Terms: | | $X_1$ | $X_2$ | $X_3$ | $X_4$ | $X_5$ | $X_6$ | $X_7$ | $X_8$ | $X_9$ | $X_{10}$ | $X_{11}$ |
| Coeffs: | | 0.04 | -0.26 | 0.09 | 0.37 | -0.04 | 0.27 | 0.10 | 0.22 | 0.42 | -0.26 | 4.07 |
| Potentials: | | 0.52 | 0.59 | 0.03 | 0.02 | -0.08 | 0.03 | -0.05 | -0.06 | 0.05 | -0.05 | 0.05 |
| Top Pairs: | | (2 1) (2 9) (2 2) (1 9) | | | | | | | | | | |
| New Term: | | $X_{12} = X_1 X_2$ | | | | | | | | | | |

Cycle 4:

| | | | | | | | | | | | | | |
|---|---|---|---|---|---|---|---|---|---|---|---|---|---|
| MSE: | 0.000 | | | | | | | | | | | | |
| Terms: | | $X_1$ | $X_2$ | $X_3$ | $X_4$ | $X_5$ | $X_6$ | $X_7$ | $X_8$ | $X_9$ | $X_{10}$ | $X_{11}$ | $X_{12}$ |
| Coeffs: | | -0.00 | -0.00 | -0.00 | 0.00 | -0.00 | 0.00 | 0.00 | 0.00 | 0.00 | -0.00 | 4.00 | 3.00 |

Solution:    $2 + 3X_1 X_2 + 4X_3 X_4 X_5$

Figure 4: A simple example of polynomial learning.

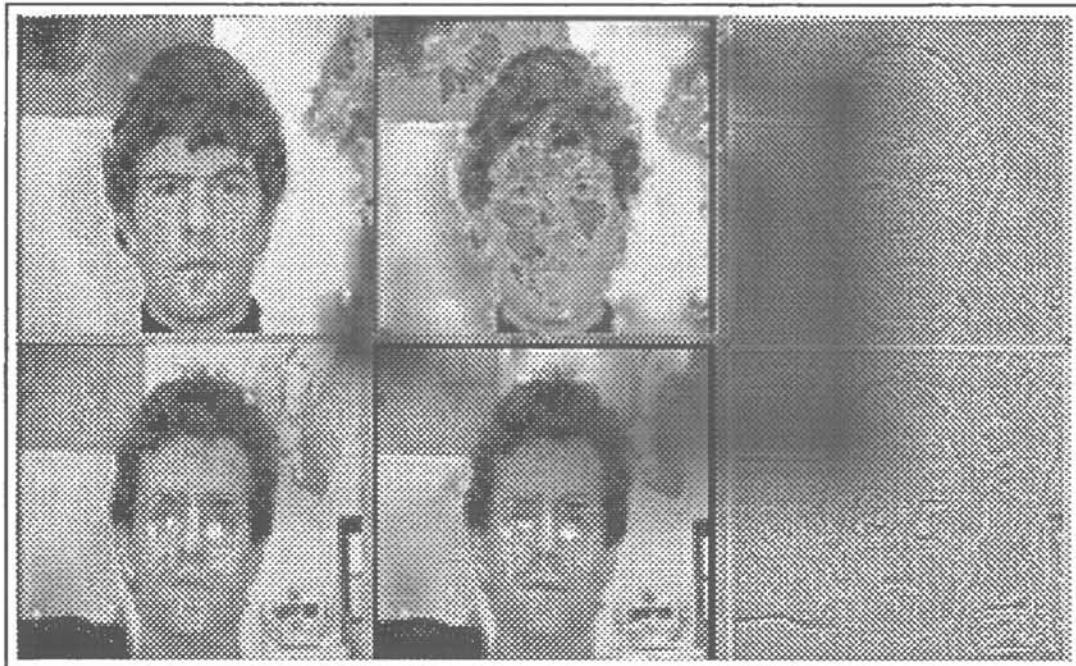

Figure 5: Original, predicted, and error images. The top row is the training image (RMS error 8.4), and the bottom row is the test image (RMS error 9.4).

$-40.1x_0 + -23.9x_1 + -5.4x_2 + -17.1x_3 +$
$(1.1x_5 + 2.4x_8 + -1.1x_2 + -1.5x_0 + -2.0x_1 + 1.3x_4 + 2.3x_6 + 3.1x_7 + -25.6)x_4 +$
$($
   $(-2.9x_9 + 3.0x_8 + -2.9x_4 + -2.8x_3 + -2.9x_2 + -1.9x_5 + -6.3x_0 + -5.2x_1 + 2.5x_6 + 6.7x_7 + 1.1)x_9 +$
   $(3.9x_8 + x_5 + 3.3x_4 + 1.6x_3 + 1.1x_2 + 2.9x_6 + 5.0x_7 + 16.1)x_8 +$
   $-2.3x_3 + -2.1x_2 + -1.6x_1 + 1.1x_4 + 2.1x_6 + 3.5x_7 + 28.6)x_5 +$
$87.1x_6 + 128.1x_7 + 80.5x_8 +$
$($
   $(-2.6x_9 + -2.4x_5 + -4.5x_0 + -3.9x_1 + 3.4x_6 + 7.3x_7 + -2.5)x_9 +$
   $21.7x_8 + -16.0x_4 + -12.1x_3 + -8.8x_2 + 31.4)x_9 +$
$2.6$

Figure 6: 55-term polynomial used to generate figure 5.

## Acknowledgments

We would like to thank Richard Brandau for his helpful comments and suggestions on an earlier draft of this paper. This report describes research done both at GTE Laboratories Incorporated, in Waltham MA, and at the laboratory of Dr. Emilio Bizzi in the department of Brain and Cognitive Sciences at MIT. T. Sanger was supported during this work by a National Defense Science and Engineering Graduate Fellowship, and by NIH grants 5R37AR26710 and 5R01NS09343 to Dr. Bizzi.

## References

Barron R. L., Mucciardi A. N., Cook F. J., Craig J. N., Barron A. R., 1984, Adaptive learning networks: Development and application in the United States of algorithms related to GMDH, In Farlow S. J., ed., *Self-Organizing Methods in Modeling*, pages 25–65, Marcel Dekker, New York.

Gabor D., 1961, A universal nonlinear filter, predictor, and simulator which optimizes itself by a learning process, *Proc. IEE*, 108B:422–438.

Grigoriev D. Y., Karpinski M., Singer M. F., 1990, Fast parallel algorithms for sparse polynomial interpolation over finite fields, *SIAM J. Computing*, 19(6):1059–1063.

Ikeda S., Ochiai M., Sawaragi Y., 1976, Sequential GMDH algorithm and its application to river flow prediction, *IEEE Trans. Systems, Man, and Cybernetics*, SMC-6(7):473–479.

Ivakhnenko A. G., 1971, Polynomial theory of complex systems, *IEEE Trans. Systems, Man, and Cybernetics*, SMC-1(4):364–378.

Sanger T. D., 1991a, Basis-function trees as a generalization of local variable selection methods for function approximation, In Lippmann R. P., Moody J. E., Touretzky D. S., ed.s, *Advances in Neural Information Processing Systems 3*, pages 700–706, Morgan Kaufmann, Proc. NIPS'90, Denver CO.

Sanger T. D., 1991b, A tree-structured adaptive network for function approximation in high dimensional spaces, *IEEE Trans. Neural Networks*, 2(2):285–293.

Sutton R. S., Matheus C. J., 1991, Learning polynomial functions by feature construction, In *Proc. Eighth Intl. Workshop on Machine Learning*, Chicago.

Widrow B., Hoff M. E., 1960, Adaptive switching circuits, In *IRE WESCON Conv. Record, Part 4*, pages 96–104.